# Assignment of Multiplicative Mixtures in Natural Images

**Odelia Schwartz**
HHMI and Salk Institute
La Jolla, CA 92014
odelia@salk.edu

**Terrence J. Sejnowski**
HHMI and Salk Institute
La Jolla, CA 92014
terry@salk.edu

**Peter Dayan**
GCNU, UCL
17 Queen Square, London
dayan@gatsby.ucl.ac.uk

## Abstract

In the analysis of natural images, Gaussian scale mixtures (GSM) have been used to account for the statistics of filter responses, and to inspire hierarchical cortical representational learning schemes. GSMs pose a critical *assignment* problem, working out which filter responses were generated by a common multiplicative factor. We present a new approach to solving this assignment problem through a probabilistic extension to the basic GSM, and show how to perform inference in the model using Gibbs sampling. We demonstrate the efficacy of the approach on both synthetic and image data.

Understanding the statistical structure of natural images is an important goal for visual neuroscience. Neural representations in early cortical areas decompose images (and likely other sensory inputs) in a way that is sensitive to sophisticated aspects of their probabilistic structure. This structure also plays a key role in methods for image processing and coding.

A striking aspect of natural images that has reflections in both top-down and bottom-up modeling is coordination across nearby locations, scales, and orientations. From a top-down perspective, this structure has been modeled using what is known as a Gaussian Scale Mixture model (GSM).[1–3] GSMs involve a *multi-dimensional Gaussian* (each dimension of which captures local structure as in a linear filter), multiplied by a spatialized collection of common hidden scale variables or *mixer variables** (which capture the coordination). GSMs have wide implications in theories of cortical receptive field development, *eg* the comprehensive bubbles framework of Hyvärinen.[4] The mixer variables provide the top-down account of two bottom-up characteristics of natural image statistics, namely the 'bowtie' statistical dependency,[5,6] and the fact that the marginal distributions of receptive field-like filters have high kurtosis.[7,8] In hindsight, these ideas also bear a close relationship with Ruderman and Bialek's multiplicative bottom-up image analysis framework[9] and statistical models for divisive gain control.[6] Coordinated structure has also been addressed in other image work,[10–14] and in other domains such as speech[15] and finance.[16]

Many approaches to the unsupervised specification of representations in early cortical areas rely on the coordinated structure.[17–21] The idea is to learn linear filters (*eg* modeling simple cells as in[22,23]), and then, based on the coordination, to find combinations of these (perhaps non-linearly transformed) as a way of finding higher order filters (*eg* complex cells). One critical facet whose specification from data is not obvious is the neighborhood arrangement, *ie* which linear filters share which mixer variables.

---

*Mixer variables are also called mutlipliers, but are *unrelated* to the scales of a wavelet.

Here, we suggest a method for finding the neighborhood based on Bayesian inference of the GSM random variables. In section 1, we consider estimating these components based on information from different-sized neighborhoods and show the modes of failure when inference is too local or too global. Based on these observations, in section 2 we propose an extension to the GSM generative model, in which the mixer variables can overlap probabilistically. We solve the neighborhood assignment problem using Gibbs sampling, and demonstrate the technique on synthetic data. In section 3, we apply the technique to image data.

## 1  GSM inference of Gaussian and mixer variables

In a simple, $n$-dimensional, version of a GSM, filter responses $\mathbf{l}$ are synthesized [†] by multiplying an $n$-dimensional Gaussian with values $\mathbf{g} = \{g_1 \ldots g_n\}$, by a common mixer variable $v$.

$$\mathbf{l} = v\mathbf{g} \tag{1}$$

We assume $\mathbf{g}$ are uncorrelated ($\sigma^2$ along diagonal of the covariance matrix). For the analytical calculations, we assume that $v$ has a Rayleigh distribution:

$$p[v] \propto [v \exp{-v^2/2}]^a \quad \text{where } 0 < a \leq 1 \text{ parameterizes the strength of the prior} \tag{2}$$

For ease, we develop the theory for $a = 1$. As is well known,[2] and repeated in figure 1(B), the marginal distribution of the resulting GSM is sparse and highly kurtotic. The joint conditional distribution of two elements $l_1$ and $l_2$, follows a bowtie shape, with the width of the distribution of one dimension increasing for larger values (both positive and negative) of the other dimension.

The inverse problem is to estimate the $n+1$ variables $g_1 \ldots g_n, v$ from the $n$ filter responses $l_1 \ldots l_n$. It is formally ill-posed, though regularized through the prior distributions. Four posterior distributions are particularly relevant, and can be derived analytically from the model:

| rv | distribution | posterior mean |
|---|---|---|
| $p[v\|l_1]$ | $\dfrac{\sqrt{\frac{\sigma}{\|l_1\|}}}{\mathcal{B}\left(\frac{1}{2}, \frac{\|l_1\|}{\sigma}\right)} \exp\left(-\frac{v^2}{2} - \frac{l_1^2}{2v^2\sigma^2}\right)$ | $\sqrt{\frac{\|l_1\|}{\sigma}} \dfrac{\mathcal{B}\left(1, \frac{\|l_1\|}{\sigma}\right)}{\mathcal{B}\left(\frac{1}{2}, \frac{\|l_1\|}{\sigma}\right)}$ |
| $p[v\|\mathbf{l}]$ | $\dfrac{\left(\frac{l}{\sigma}\right)^{\frac{1}{2}(n-2)}}{\mathcal{B}\left(1-\frac{n}{2}, \frac{l}{\sigma}\right)} v^{-(n-1)} \exp\left(-\frac{v^2}{2} - \frac{l^2}{2v^2\sigma^2}\right)$ | $\sqrt{\frac{l}{\sigma}} \dfrac{\mathcal{B}\left(\frac{3}{2} - \frac{n}{2}, \frac{l}{\sigma}\right)}{\mathcal{B}\left(1 - \frac{n}{2}, \frac{l}{\sigma}\right)}$ |
| $p[\|g_1\|\|l_1]$ | $\dfrac{\sqrt{\sigma\|l_1\|}}{\mathcal{B}\left(-\frac{1}{2}, \frac{\|l_1\|}{\sigma}\right)} \dfrac{1}{g_1^2} \exp\left(-\frac{g_1^2}{2\sigma^2} - \frac{l_1^2}{2g_1^2}\right)$ | $\sigma\sqrt{\frac{\|l_1\|}{\sigma}} \dfrac{\mathcal{B}\left(0, \frac{\|l_1\|}{\sigma}\right)}{\mathcal{B}\left(-\frac{1}{2}, \frac{\|l_1\|}{\sigma}\right)}$ |
| $p[\|g_1\|\|\mathbf{l}]$ | $\dfrac{\sqrt{\sigma\|l_1\|}\left(\frac{\|l_1\|}{l}\right)^{\frac{1}{2}(2-n)}}{\mathcal{B}\left(\frac{n}{2}-1, \frac{l}{\sigma}\right)} g_1^{(n-3)} \exp\left(-\frac{g_1^2}{2\sigma^2}\frac{l^2}{l_1^2} - \frac{l_1^2}{2g_1^2}\right)$ | $\sigma\sqrt{\frac{\|l_1\|}{\sigma}} \sqrt{\frac{\|l_1\|}{l}} \dfrac{\mathcal{B}\left(\frac{n}{2}-\frac{1}{2}, \frac{l}{\sigma}\right)}{\mathcal{B}\left(\frac{n}{2}-1, \frac{l}{\sigma}\right)}$ |

where $\mathcal{B}(n, x)$ is the modified Bessel function of the second kind (see also[24]), $l = \sqrt{\sum_i l_i^2}$ and $g_i$ is forced to have the same sign as $l_i$, since the mixer variables are always positive. Note that $p[v|l_1]$ and $p[g_1|l_1]$ (rows 1,3) are local estimates, while $p[v|\mathbf{l}]$ and $p[g|\mathbf{l}]$ (rows 2,4) are estimates according to filter outputs $\{l_1 \ldots l_n\}$. The posterior $p[v|\mathbf{l}]$ has also been estimated numerically in noise removal for other mixer priors, by Portilla *et al*[25]

The full GSM specifies a hierarchy of mixer variables. Wainwright[2] considered a pre-specified tree-based hierarhical arrangement. In practice, for natural sensory data, given a heterogeneous collection of $l_i$, it is advantageous to learn the hierachical arrangement from examples. In an approach related to that of the GSM, Karklin and Lewicki[19] suggested

---

[†]We describe the $\mathbf{l}$ as being filter responses even in the synthetic case, to facilitate comparison with images.

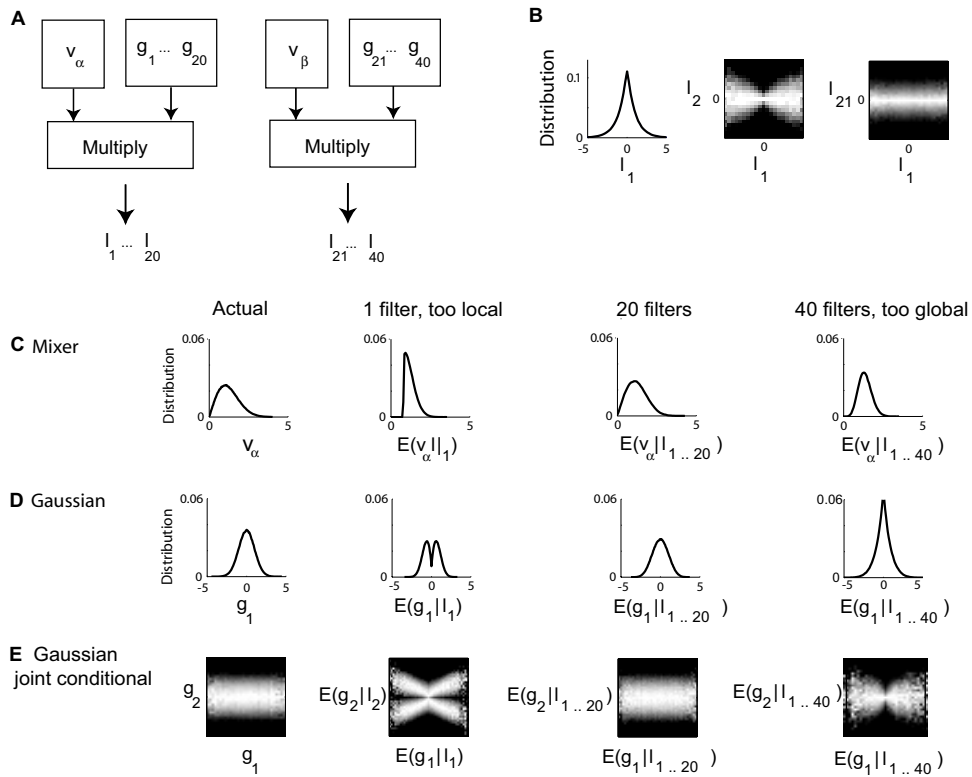

**Figure 1: A** Generative model: each filter response is generated by multiplying its Gaussian variable by either mixer variable $v_\alpha$, or mixer variable $v_\beta$. **B** Marginal and joint conditional statistics (bowties) of sample synthetic filter responses. For the joint conditional statistics, intensity is proportional to the bin counts, except that each column is independently re-scaled to fill the range of intensities. **C-E** Left: actual distributions of mixer and Gaussian variables; other columns: estimates based on different numbers of filter responses. **C** Distribution of estimate of the mixer variable $v_\alpha$. Note that mixer variable values are by definition positive. **D** Distribution of estimate of one of the Gaussian variables, $g_1$. **E** Joint conditional statistics of the estimates of Gaussian variables $g_1$ and $g_2$.

generating log mixer values for all the filters and learning the linear combinations of a smaller collection of underlying values. Here, we consider the problem in terms of multiple mixer variables, with the linear filters being clustered into groups that share a single mixer. This poses a critical assignment problem of working out which filter responses share which mixer variables. We first study this issue using synthetic data in which two groups of filter responses $l_1 \ldots l_{20}$ and $l_{21} \ldots l_{40}$ are generated by two mixer variables $v_\alpha$ and $v_\beta$ (figure 1). We attempt to infer the components of the GSM model from the synthetic data.

Figure 1C;D shows the empirical distributions of estimates of the conditional means of a mixer variable $E(v_\alpha|\{l\})$ and one of the Gaussian variables $E(g_1|\{l\})$ based on different assumed assignments. For estimation based on too few filter responses, the estimates do not well match the actual distributions. For example, for a local estimate based on a single filter response, the Gaussian estimate peaks away from zero. For assignments including more filter responses, the estimates become good. However, inference is also compromised if the estimates for $v_\alpha$ are too global, including filter responses actually generated from $v_\beta$ (C and D, last column). In (E), we consider the joint conditional statistics of two components, each

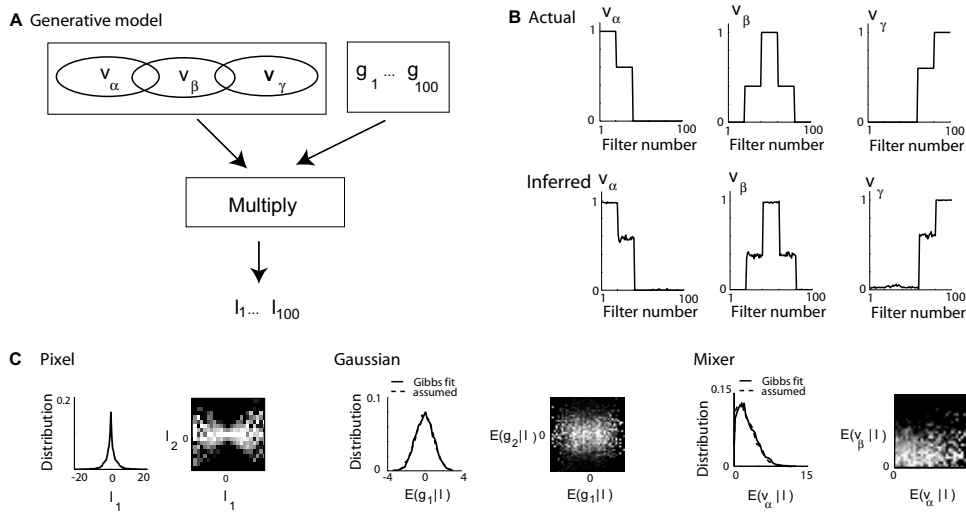

**Figure 2: A** Generative model in which each filter response is generated by multiplication of its Gaussian variable by a mixer variable. The mixer variable, $v_\alpha$, $v_\beta$, or $v_\gamma$, is chosen probabilistically upon each filter response sample, from a Rayleigh distribution with $a = .1$. **B** Top: actual probability of filter associations with $v_\alpha$, $v_\beta$, and $v_\gamma$; Bottom: Gibbs estimates of probability of filter associations corresponding to $v_\alpha$, $v_\beta$, and $v_\gamma$. **C** Statistics of generated filter responses, and of Gaussian and mixer estimates from Gibbs sampling.

estimating their respective $g_1$ and $g_2$. Again, as the number of filter responses increases, the estimates improve, provided that they are taken from the right group of filter responses with the same mixer variable. Specifically, the mean estimates of $g_1$ and $g_2$ become more independent (E, third column). Note that for estimations based on a single filter response, the joint conditional distribution of the Gaussian appears correlated rather than independent (E, second column); for estimation based on too many filter responses (40 in this example), the joint conditional distribution of the Gaussian estimates shows a dependent (rather than independent) bowtie shape (E, last column). Mixer variable joint statistics also deviate from the actual when the estimations are too local or global (not shown).

We have observed qualitatively similar statistics for estimation based on coefficients in natural images. Neighborhood size has also been discussed in the context of the quality of noise removal, assuming a GSM model.[26]

## 2   Neighborhood inference: solving the assignment problem

The plots in figure 1 suggest that it should be possible to infer the assignments, *ie* work out which filter responses share common mixers, by learning from the statistics of the resulting joint dependencies. Hard assignment problems (in which each filter response pays allegiance to just one mixer) are notoriously computationally brittle. Soft assignment problems (in which there is a probabilistic relationship between filter responses and mixers) are computationally better behaved. Further, real world stimuli are likely better captured by the possibility that filter responses are coordinated in somewhat different collections in different images.

We consider a richer, mixture GSM as a generative model (Figure 2). To model the generation of filter responses $l_i$ for a single image patch, we multiply each Gaussian variable $g_i$ by a single mixer variable from the set $v_1 \ldots v_m$. We assume that $g_i$ has *association* probabil-

ity $p_{ij}$ (satisfying $\sum_j p_{ij} = 1, \forall i$) of being assigned to mixer variable $v_j$. The assignments are assumed to be made independently for each patch. We use $s_i \in \{1, 2, \ldots m\}$ for the assignments:

$$l_i = g_i v_{s_i} \tag{3}$$

Inference and learning in this model proceeds in two stages, according to the expectation maximization algorithm. First, given a filter response $l_i$, we use Gibbs sampling for the E phase to find possible appropriate (posterior) assignments. Williams et al.[27] suggested using Gibbs sampling to solve a similar assignment problem in the context of dynamic tree models. Second, for the M phase, given the collection of assignments across multiple filter responses, we update the association probabilities $p_{ij}$. Given sample mixer assignments, we can estimate the Gaussian and mixer components of the GSM using the table of section 1, but restricting the filter response samples just to those associated with each mixer variable.

We tested the ability of this inference method to find the associations in the probabilistic mixer variable synthetic example shown in figure 2, (A,B). The true generative model specifies probabilistic overlap of 3 mixer variables. We generated 5000 samples for each filter according to the generative model. We ran the Gibbs sampling procedure, setting the number of possible neighborhoods to 5 (e.g., $> 3$); after 500 iterations the weights converged near to the proper probabilities. In (B, top), we plot the actual probability distributions for the filter associations with each of the mixer variables. In (B, bottom), we show the estimated associations: the three non-zero estimates closely match the actual distributions; the other two estimates are zero (not shown). The procedure consistently finds correct associations even in larger examples of data generated with up to 10 mixer variables. In (C) we show an example of the actual and estimated distributions of the mixer and Gaussian components of the GSM. Note that the joint conditional statistics of both mixer and Gaussian are independent, since the variables were generated as such in the synthetic example. The Gibbs procedure can be adjusted for data generated with different parameters $a$ of equation 2, and for related mixers,[2] allowing for a range of image coefficient behaviors.

## 3   Image data

Having validated the inference model using synthetic data, we turned to natural images. We derived linear filters from a multi-scale oriented steerable pyramid,[28] with 100 filters, at 2 preferred orientations, 25 non-overlapping spatial positions (with spatial subsampling of 8 pixels), and two phases (quadrature pairs), and a single spatial frequency peaked at $1/6$ cycles/pixel. The image ensemble is 4 images from a standard image compression database (boats, goldhill, plant leaves, and mountain) and 4000 samples.

We ran our method with the same parameters as for synthetic data, with 7 possible neighborhoods and Rayleigh parameter $a = .1$ (as in figure 2). Figure 3 depicts the association weights $p_{ij}$ of the coefficients for each of the obtained mixer variables. In (A), we show a schematic (template) of the association representation that will follow in (B, C) for the actual data. Each mixer variable neighborhood is shown for coefficients of two phases and two orientations along a spatial grid (one grid for each phase). The neighborhood is illustrated via the probability of each coefficient to be generated from a given mixer variable. For the first two neighborhoods (B), we also show the image patches that yielded the maximum log likelihood of $P(v|patch)$. The first neighborhood (in B) prefers vertical patterns across most of its "receptive field", while the second has a more localized region of horizontal preference. This can also be seen by averaging the 200 image patches with the maximum log likelihood. Strikingly, all the mixer variables group together two phases of quadrature pair (B, C). Quadrature pairs have also been extracted from cortical data, and are the components of ideal complex cell models. Another tendency is to group

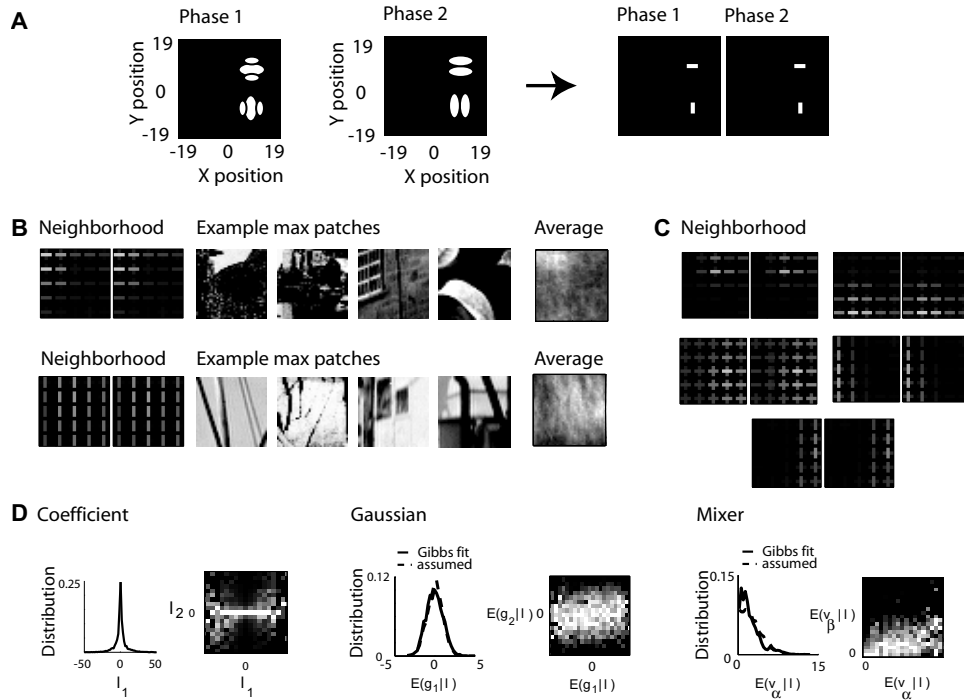

**Figure 3: A** Schematic of the mixer variable neighborhood representation. The probability that each coefficient is associated with the mixer variable ranges from 0 (black) to 1 (white). Left: Vertical and horizontal filters, at two orientations, and two phases. Each phase is plotted separately, on a 38 by 38 pixel spatial grid. Right: summary of representation, with filter shapes replaced by oriented lines. Filters are approximately 6 pixels in diameter, with the spacing between filters 8 pixels. **B** First two image ensemble neighborhoods obtained from Gibbs sampling. Also shown, are four $38 \times 38$ pixel patches that had the maximum log likelihood of $P(v|patch)$, and the average of the first 200 maximal patches. **C** Other image ensemble neighborhoods. **D** Statistics of representative coefficients of two spatially displaced vertical filters, and of inferred Gaussian and mixer variables.

orientations across space. The phase and iso-orientation grouping bear some interesting similarity to other recent suggestions;[17, 18] as do the maximal patches.[19] Wavelet filters have the advantage that they can span a wider spatial extent than is possible with current ICA techniques, and the analysis of parameters such as phase grouping is more controlled. We are comparing the analysis with an ICA first-stage representation, which has other obvious advantages. We are also extending the analysis to correlated wavelet filters;[25] and to simulations with a larger number of neighborhoods.

From the obtained associations, we estimated the mixer and Gaussian variables according to our model. In (D) we show representative statistics of the coefficients and of the inferred variables. The learned distributions of Gaussian and mixer variables are quite close to our assumptions. The Gaussian estimates exhibit joint conditional statistics that are roughly independent, and the mixer variables are weakly dependent.

We have thus far demonstrated neighborhood inference for an image ensemble, but it is also interesting and perhaps more intuitive to consider inference for particular images or image classes. In figure 4 (A-B) we demonstrate example mixer variable neighborhoods derived from learning patches of a zebra image (Corel CD-ROM). As before, the neighborhoods are composed of quadrature pairs; however, the spatial configurations are richer and have

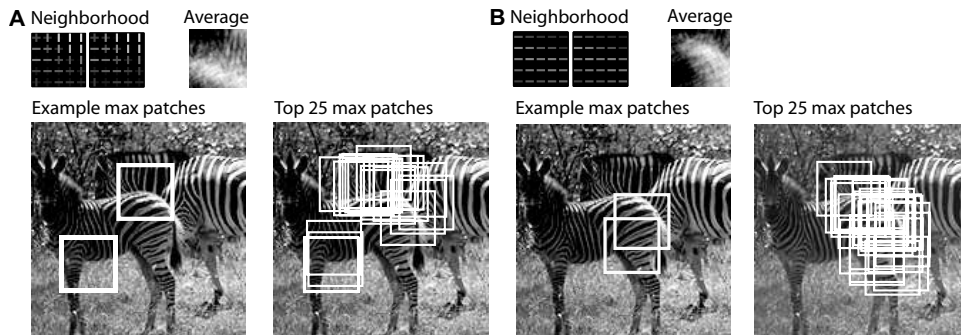

**Figure 4:** Example of Gibbs on Zebra image. Image is $151\times151$ pixels, and each spatial neighborhood spans $38\times38$ pixels. **A, B** Example mixer variable neighborhoods. Left: example mixer variable neighborhood, and average of 200 patches that yielded the maximum likelihood of $P(v|patch)$. Right: Image and marked on top of it example patches that yielded the maximum likelihood of $P(v|patch)$.

not been previously reported with unsupervised hierarchical methods: for example, in (A), the mixture neighborhood captures a horizontal-bottom/vertical-top spatial configuration. This appears particularly relevant in segmenting regions of the front zebra, as shown by marking in the image the patches $i$ that yielded the maximum log likelihood of $P(v|patch)$. In (B), the mixture neighborhood captures a horizontal configuration, more focused on the horizontal stripes of the front zebra. This example demonstrates the logic behind a probabilistic mixture: coefficients corresponding to the bottom horizontal stripes might be linked with top vertical stripes (A) or to more horizontal stripes (B).

## 4   Discussion

Work on the study of natural image statistics has recently evolved from issues about scale-space hierarchies, wavelets, and their ready induction through unsupervised learning models (loosely based on cortical development) towards the coordinated statistical structure of the wavelet components. This includes bottom-up (*eg* bowties, hierarchical representations such as complex cells) and top-down (*eg* GSM) viewpoints. The resulting new insights inform a wealth of models and ideas and form the essential backdrop for the work in this paper. They also link to impressive engineering results in image coding and processing.

A most critical aspect of an hierarchical representational model is the way that the structure of the hierarchy is induced. We addressed the hierarchy question using a novel extension to the GSM generative model in which mixer variables (at one level of the hierarchy) enjoy probabilistic assignments to filter responses (at a lower level). We showed how these assignments can be learned (using Gibbs sampling), and illustrated some of their attractive properties using both synthetic and a variety of image data. We grounded our method firmly in Bayesian inference of the posterior distributions over the two classes of random variables in a GSM (mixer and Gaussian), placing particular emphasis on the interplay between the generative model and the statistical properties of its components.

An obvious question raised by our work is the neural correlate of the two different posterior variables. The Gaussian variable has characteristics resembling those of the *output* of divisively normalized simple cells;[6] the mixer variable is more obviously related to the *output* of quadrature pair neurons (such as orientation energy or motion energy cells, which may also be divisively normalized). How these different information sources may subsequently be used is of great interest.

**Acknowledgements** This work was funded by the HHMI (OS, TJS) and the Gatsby Charitable Foundation (PD). We are very grateful to Patrik Hoyer, Mike Lewicki, Zhaoping Li, Simon Osindero, Javier Portilla and Eero Simoncelli for discussion.

## References

[1] D Andrews and C Mallows. Scale mixtures of normal distributions. *J. Royal Stat. Soc.*, 36:99–102, 1974.

[2] M J Wainwright and E P Simoncelli. Scale mixtures of Gaussians and the statistics of natural images. In S. A. Solla, T. K. Leen, and K.-R. Müller, editors, *Adv. Neural Information Processing Systems*, volume 12, pages 855–861, Cambridge, MA, May 2000. MIT Press.

[3] M J Wainwright, E P Simoncelli, and A S Willsky. Random cascades on wavelet trees and their use in modeling and analyzing natural imagery. *Applied and Computational Harmonic Analysis*, 11(1):89–123, July 2001. Special issue on wavelet applications.

[4] A Hyvärinen, J Hurri, and J Vayrynen. Bubbles: a unifying framework for low-level statistical properties of natural image sequences. *Journal of the Optical Society of America A*, 20:1237–1252, May 2003.

[5] R W Buccigrossi and E P Simoncelli. Image compression via joint statistical characterization in the wavelet domain. *IEEE Trans Image Proc*, 8(12):1688–1701, December 1999.

[6] O Schwartz and E P Simoncelli. Natural signal statistics and sensory gain control. *Nature Neuroscience*, 4(8):819–825, August 2001.

[7] D J Field. Relations between the statistics of natural images and the response properties of cortical cells. *J. Opt. Soc. Am. A*, 4(12):2379–2394, 1987.

[8] H Attias and C E Schreiner. Temporal low-order statistics of natural sounds. In M Jordan, M Kearns, and S Solla, editors, *Adv in Neural Info Processing Systems*, volume 9, pages 27–33. MIT Press, 1997.

[9] D L Ruderman and W Bialek. Statistics of natural images: Scaling in the woods. *Phys. Rev. Letters*, 73(6):814–817, 1994.

[10] C Zetzsche, B Wegmann, and E Barth. Nonlinear aspects of primary vision: Entropy reduction beyond decorrelation. In *Int'l Symposium, Society for Information Display*, volume XXIV, pages 933–936, 1993.

[11] J Huang and D Mumford. Statistics of natural images and models. In *CVPR*, page 547, 1999.

[12] J. Romberg, H. Choi, and R. Baraniuk. Bayesian wavelet domain image modeling using hidden Markov trees. In *Proc. IEEE Int'l Conf on Image Proc*, Kobe, Japan, October 1999.

[13] A Turiel, G Mato, N Parga, and J P Nadal. The self-similarity properties of natural images resemble those of turbulent flows. *Phys. Rev. Lett.*, 80:1098–1101, 1998.

[14] J Portilla and E P Simoncelli. A parametric texture model based on joint statistics of complex wavelet coefficients. *Int'l Journal of Computer Vision*, 40(1):49–71, 2000.

[15] Helmut Brehm and Walter Stammler. Description and generation of spherically invariant speech-model signals. *Signal Processing*, 12:119–141, 1987.

[16] T Bollersley, K Engle, and D Nelson. ARCH models. In B Engle and D McFadden, editors, *Handbook of Econometrics V*. 1994.

[17] A Hyvärinen and P Hoyer. Emergence of topography and complex cell properties from natural images using extensions of ICA. In S. A. Solla, T. K. Leen, and K.-R. Müller, editors, *Adv. Neural Information Processing Systems*, volume 12, pages 827–833, Cambridge, MA, May 2000. MIT Press.

[18] P Hoyer and A Hyvärinen. A multi-layer sparse coding network learns contour coding from natural images. *Vision Research*, 42(12):1593–1605, 2002.

[19] Y Karklin and M S Lewicki. Learning higher-order structures in natural images. *Network: Computation in Neural Systems*, 14:483–499, 2003.

[20] W Laurenz and T Sejnowski. Slow feature analysis: Unsupervised learning of invariances. *Neural Computation*, 14(4):715–770, 2002.

[21] C Kayser, W Einhäuser, O Dümmer, P König, and K P Körding. Extracting slow subspaces from natural videos leads to complex cells. In G Dorffner, H Bischof, and K Hornik, editors, *Proc. Int'l Conf. on Artificial Neural Networks (ICANN-01)*, pages 1075–1080, Vienna, Aug 2001. Springer-Verlag, Heidelberg.

[22] B A Olshausen and D J Field. Emergence of simple-cell receptive field properties by learning a sparse factorial code. *Nature*, 381:607–609, 1996.

[23] A J Bell and T J Sejnowski. The 'independent components' of natural scenes are edge filters. *Vision Research*, 37(23):3327–3338, 1997.

[24] U Grenander and A Srivastava. Probability models for clutter in natural images. *IEEE Trans. on Patt. Anal. and Mach. Intel.*, 23:423–429, 2002.

[25] J Portilla, V Strela, M Wainwright, and E Simoncelli. Adaptive Wiener denoising using a Gaussian scale mixture model in the wavelet domain. In *Proc 8th IEEE Int'l Conf on Image Proc*, pages 37–40, Thessaloniki, Greece, Oct 7-10 2001. IEEE Computer Society.

[26] J Portilla, V Strela, M Wainwright, and E P Simoncelli. Image denoising using a scale mixture of Gaussians in the wavelet domain. *IEEE Trans Image Processing*, 12(11):1338–1351, November 2003.

[27] C K I Williams and N J Adams. Dynamic trees. In M. S. Kearns, S. A. Solla, and D. A. Cohn, editors, *Adv. Neural Information Processing Systems*, volume 11, pages 634–640, Cambridge, MA, 1999. MIT Press.

[28] E P Simoncelli, W T Freeman, E H Adelson, and D J Heeger. Shiftable multi-scale transforms. *IEEE Trans Information Theory*, 38(2):587–607, March 1992. Special Issue on Wavelets.